# Non-Parametric Modeling of Partially Ranked Data

**Guy Lebanon**
Department of Statistics, and
School of Elec. and Computer Engineering
Purdue University - West Lafayette, IN
lebanon@stat.purdue.edu

**Yi Mao**
School of Elec. and Computer Engineering
Purdue University - West Lafayette, IN
ymao@ecn.purdue.edu

## Abstract

Statistical models on full and partial rankings of $n$ items are often of limited practical use for large $n$ due to computational consideration. We explore the use of non-parametric models for partially ranked data and derive efficient procedures for their use for large $n$. The derivations are largely possible through combinatorial and algebraic manipulations based on the lattice of partial rankings. In particular, we demonstrate for the first time a non-parametric coherent and consistent model capable of efficiently aggregating partially ranked data of different types.

## 1   Introduction

Rankers such as humans, search engines, and classifiers, output full or partial rankings representing preference relations over $n$ items. The absence of numeric scores or the lack of calibration between existing numeric scores output by the rankers necessitates modeling rankings rather than numeric scores. To effectively analyze ranked data, a statistical model has the following desiderata.

**(1)** Handle efficiently a very large number of items $n$ by reverting to partial rather than full rankings.

**(2)** Probability assignment to full and partial rankings should be coherent and contradiction-free.

**(3)** Conduct inference based on training data consisting of partial rankings of different types.

**(4)** Correct retrieval of the underlying process as training data increases (statistical consistency).

**(5)** In the case of large $n$ convergence of the estimator to the underlying process can be extremely slow for fully ranked data but should be much faster when restricted to simpler partial rankings.

In this paper, we present a model achieving the above requirements without any parametric assumptions on the underlying generative process. The model is based on the non-parametric Parzen window estimator with a Mallows kernel on permutations. By considering partial rankings as censored data we are able to define the model on both full and partial rankings in a coherent and contradiction-free manner. Furthermore, we are able to estimate the underlying structure based on data containing partial rankings of different types. We demonstrate computational efficiency for partial rankings, even in the case of a very large $n$, by exploiting the combinatorial and algebraic structure of the lattice of partial rankings. We start below by reviewing basic concepts concerning partially ranked data (see [1] for further details) and the Mallows model and then proceed to define our non-parametric estimator. We conclude by demonstrating computational efficiency and some experiments.

## 2   Permutations and Cosets

A permutation $\pi$ is a bijective function $\pi : \{1, \ldots, n\} \rightarrow \{1, \ldots, n\}$ associating with each item $i \in \{1, \ldots, n\}$ a rank $\pi(i) \in \{1, \ldots, n\}$. In other words, $\pi(i)$ denotes the rank given to item $i$

and $\pi^{-1}(i)$ denotes the item assigned to rank $i$. We denote a permutation $\pi$ using the following vertical bar notation $\pi^{-1}(1)|\pi^{-1}(2)|\cdots|\pi^{-1}(n)$. For example, the permutation $\pi(1) = 2, \pi(2) = 3, \pi(3) = 1$ would be denoted as $3|1|2$. In this notation the numbers correspond to items and the locations of the items in their corresponding compartments correspond to their ranks. The collection of all permutations of $n$ items forms the non-Abelian symmetric group of order $n$, denoted by $\mathfrak{S}_n$, using function composition as the group operation $\pi\sigma = \pi \circ \sigma$. We denote the identity permutation by $e$. The concept of inversions and the result below, taken from [7], will be of great use later on.

**Definition 1.** *The inversion set of a permutation $\pi$ is the set of pairs*

$$U(\pi) \stackrel{\text{def}}{=} \{(i,j) \,:\, i < j, \ \pi(i) > \pi(j)\} \subset \{1,\dots,n\} \times \{1,\dots,n\}$$

*whose cardinality is denoted by $i(\pi) \stackrel{\text{def}}{=} |U(\pi)|$.*

For example, $i(e) = |\emptyset| = 0$, and $i(3|2|1|4) = |\{(1,2),(1,3),(2,3)\}| = 3$.

**Proposition 1** (e.g., [7]). *The map $\pi \mapsto U(\pi)$ is a bijection.*

When $n$ is large, the enormous number of permutations raises difficulties in using the symmetric group for modeling rankings. A reasonable solution is achieved by considering partial rankings which correspond to cosets of the symmetric group. For example, the subgroup of $\mathfrak{S}_n$ consisting of all permutations that fix the top $k$ positions is denoted $\mathfrak{S}_{1,\dots,1,n-k} = \{\pi \in \mathfrak{S}_n \,:\, \pi(i) = i, \ i = 1,\dots,k\}$. The right coset $\mathfrak{S}_{1,\dots,1,n-k}\pi = \{\sigma\pi \,:\, \sigma \in \mathfrak{S}_{1,\dots,1,n-k}\}$ is the set of permutations consistent with the ordering of $\pi$ on the $k$ top-ranked items. It may thus be interpreted as a partial ranking of the top $k$ items, that does not contain any information concerning the relative ranking of the bottom $n-k$ items. The set of all such partial rankings forms the quotient space $\mathfrak{S}_n/\mathfrak{S}_{1,\dots,1,n-k}$. Figure 1 (left) displays the set of permutations that corresponds to a partial ranking of the top 2 out of 4 items. We generalize this concept to arbitrary partial rankings using the concept of composition.

**Definition 2.** *A composition of $n$ is a sequence $\gamma = (\gamma_1,\dots,\gamma_r)$ of positive integers whose sum is $n$.*

Note that in contrast to a partition, in a composition the order of the integers matters. A composition $\gamma = (\gamma_1,\dots,\gamma_r)$ corresponds to a partial ranking with $\gamma_1$ items in the first position, $\gamma_2$ items in the second position and so on. For such a partial ranking it is known that the first set of $\gamma_1$ items are to be ranked before the second set of $\gamma_2$ items etc., but no further information is conveyed about the orderings within each set. The partial ranking $\mathfrak{S}_{1,\dots,1,n-k}\pi$ of the top $k$ items is a special case corresponding to $\gamma = (1,\dots,1,n-k)$. More formally, let $N_1 = \{1,\dots,\gamma_1\}, N_2 = \{\gamma_1+1,\dots,\gamma_1+\gamma_2\},\cdots,N_r = \{\gamma_1+\cdots+\gamma_{r-1}+1,\dots,n\}$. Then the subgroup $\mathfrak{S}_\gamma$ contains all permutations $\pi$ for which the set equalities $\pi(N_i) = N_i, \forall i$ holds (all permutations that only permute within each $N_i$). A partial ranking of type $\gamma$ is equivalent to a coset $\mathfrak{S}_\gamma\pi = \{\sigma\pi : \sigma \in \mathfrak{S}_\gamma, \pi \in \mathfrak{S}_n\}$ and the set of such partial rankings forms the quotient space $\mathfrak{S}_n/\mathfrak{S}_\gamma$.

The vertical bar notation described above is particularly convenient for denoting partial rankings. We list items $1,\dots,n$ separated by vertical bars, indicating that items on the left side of each vertical bar are preferred to (ranked higher than) items on the right side of the bar. For example, the partial ranking displayed in Figure 1 (left) is denoted by $3|1|2,4$. In the notation above, the ordering of items not separated by a vertical line is meaningless, and for consistency we use the conventional ordering e.g., $1|2,3|4$ rather than $1|3,2|4$.

The set of all partial rankings

$$\mathfrak{W}_n \stackrel{\text{def}}{=} \{\mathfrak{S}_\gamma\pi : \pi \in \mathfrak{S}_n, \forall\gamma\} \tag{1}$$

which includes all full rankings $\pi \in \mathfrak{S}_n$, is a subset of all possible partial orders on $\{1,\dots,n\}$. While the formalism of partial rankings in $\mathfrak{W}_n$ cannot realize all partial orderings, it is sufficiently powerful to include many useful naturally occurring orderings as special cases. Furthermore, as demonstrated in later sections, it enables simplification of the otherwise overwhelming computational difficulty. Special cases include the following partial rankings.

- $\pi \in \mathfrak{S}_n$ corresponds to permutation or a full ordering e.g. $3|2|4|1$.
- $\mathfrak{S}_{1,n-1}\pi$ e.g. $3|1,2,4$, corresponds to selection of the top alternative such as a multiclass classification.
- $\mathfrak{S}_{1,\dots,1,n-k}\pi$ e.g. $1|3|2,4$, corresponds to top $k$ ordering such as the ranked list of top $k$ webpages output by search engines.

- $\mathfrak{S}_{k,n-k}\pi$ e.g. $1, 2, 4|3, 5$, corresponds to a more preferred and a less preferred dichotomy such as a multilabel classification.

In the cases above, we often have a situation where $n$ is large (or even approaching infinity as in the third example above) but $k$ is of manageable size. Traditionally, data from each one of the special cases above was modeled using different tools and was considered fundamentally different. That problem was aggravated as different special cases were usually handled by different communities such as statistics, computer science, and information retrieval.

In constructing a statistical model on permutations or cosets, it is essential to relate one permutation to another. We do this using a distance function on permutations $d : \mathfrak{S}_n \times \mathfrak{S}_n \to \mathbb{R}$ that satisfies the usual metric function properties, and in addition is invariant under item relabeling or right action of the symmetric group [1] $d(\pi, \sigma) = d(\pi\tau, \sigma\tau) \ \forall \ \pi, \sigma, \tau \in \mathfrak{S}_n$. There have been many propositions for such right-invariant distance functions, the most popular of them being Kendall's tau [3]

$$d(\pi, \sigma) = \sum_{i=1}^{n-1} \sum_{l > i} I(\pi\sigma^{-1}(i) - \pi\sigma^{-1}(l)) \tag{2}$$

where $I(x) = 1$ for $x > 0$ and $I(x) = 0$ otherwise. Kendall's tau $d(\pi, \sigma)$ can be interpreted as the number of pairs of items for which $\pi$ and $\sigma$ have opposing orderings (called disconcordant pairs) or the minimum number of adjacent transpositions needed to bring $\pi^{-1}$ to $\sigma^{-1}$ (adjacent transposition flips a pair of items having adjacent ranks). By right invariance, $d(\pi, \sigma) = d(\pi\sigma^{-1}, e)$ which, for Kendall's tau equals the number of inversions $i(\pi\sigma^{-1})$. This is an important observation that will allow us to simplify many expressions concerning Kendall's tau using the theory of permutation inversions from the combinatorics literature.

## 3 The Mallows Model and its Extension to Partial Rankings

The Mallows model [5] is a simple model on permutations based on Kendall's tau distance using a location parameter $\kappa$ and a spread parameter $c$ (which we often treat as a constant)

$$p_\kappa(\pi) = \exp\left(-c\,d(\pi, \kappa) - \log \psi(c)\right) \qquad \pi, \kappa \in \mathfrak{S}_n \quad c \in \mathbb{R}_+. \tag{3}$$

The normalization term $\psi$ doesn't depend on $\kappa$ and has the closed form

$$\psi(c) = \sum_{\pi \in \mathfrak{S}_n} e^{-c\,d(\pi, \kappa)} = (1 + e^{-c})(1 + e^{-c} + e^{-2c}) \cdots (1 + e^{-c} + \cdots + e^{-(n-1)c}) \tag{4}$$

as shown by the fact that $d(\pi, \sigma) = i(\pi\sigma^{-1})$ and the following proposition.

**Proposition 2** (e.g., [7]). *For $q > 0$, $\sum_{\pi \in \mathfrak{S}_n} q^{i(\pi)} = \prod_{j=1}^{n-1} \sum_{k=0}^{j} q^k$.*

Model (3) has been motivated on axiomatic grounds by Mallows and has been a major focus of statistical modeling on permutations. A natural extension to partially ranked data is to consider a partial ranking as censored data equivalent to the set of permutations in its related coset:

$$p_\kappa(\mathfrak{S}_\gamma \pi) \stackrel{\text{def}}{=} \sum_{\tau \in \mathfrak{S}_\gamma \pi} p_\kappa(\tau) = \psi^{-1}(c) \sum_{\tau \in \mathfrak{S}_\gamma \pi} \exp\left(-c\,d(\tau, \kappa)\right). \tag{5}$$

Fligner and Verducci [2] have shown that in the case of $\gamma = (1, \dots, 1, n - k)$ the above summation has a closed form expression. However, the apparent absence of a closed form formula for more general partial rankings prevented the widespread use of the above model for large $n$ and encouraged more ad-hoc and heuristic models [1, 6]. This has become especially noticeable due to a new surge of interest, especially in the computer science community, in partial ranking models for large $n$. The ranking lattice presented next enables extending Fligner and Verducci's closed form to a more general setting which is critical to the practicality of our non-parametric estimator.

## 4 The Ranking Lattice

Partial rankings $\mathfrak{S}_\gamma \pi$ relate to each other in a natural way by expressing more general, more specific or inconsistent ordering. We define below the concepts of partially ordered sets and lattices and then relate them to partial rankings by considering the set of partial rankings $\mathfrak{W}_n$ as a lattice. Some of the definitions below are taken from [7], where a thorough introduction to posets can be found.

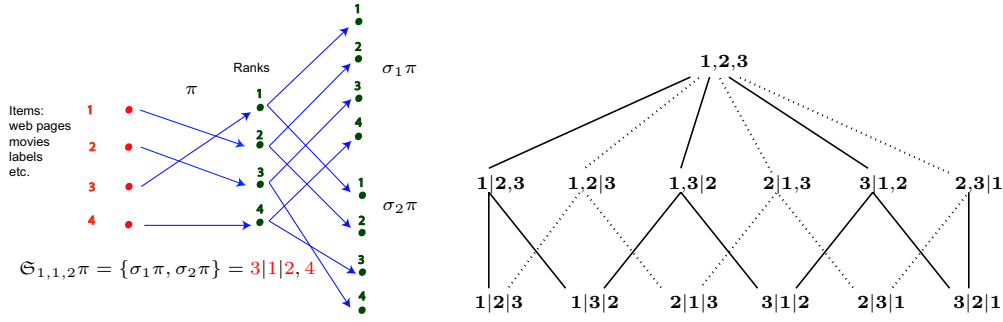

Figure 1: A partial ranking corresponds to a coset or a set of permutations (left). The Hasse diagram of $\mathfrak{W}_3$. Some lines are dotted for 3D visualization purposes (right).

**Definition 3.** *A partially ordered set or poset $(Q, \preceq)$, is a set $Q$ endowed with a binary relation $\preceq$ satisfying $\forall x, y, z \in Q$ (i) reflexibility: $x \preceq x$, (ii) anti-symmetry: $x \preceq y$ and $y \preceq x \Rightarrow x = y$, and (iii) transitivity: $x \preceq y$ and $y \preceq z \Rightarrow x \preceq z$.*

We write $x \prec y$ when $x \preceq y$ and $x \neq y$. We say that $y$ covers $x$ when $x \prec y$ and there is no $z \in Q$ such that $x \prec z \prec y$. A finite poset is completely described by its covering relation. The planar Hasse diagram of $(Q, \preceq)$ is the graph connecting the elements of $Q$ as nodes using edges that correspond to the covering relation. An additional requirement is that if $y$ covers $x$ then $y$ is drawn higher than $x$. Two elements $x, y$ are comparable if $x \preceq y$ or $y \preceq x$ and otherwise are incomparable. The set of partial rankings $\mathfrak{W}_n$ defined in (1) is naturally endowed with the partial order of ranking refinement i.e. $\pi \prec \sigma$ if $\pi$ refines $\sigma$ or alternatively if we can get from $\pi$ to $\sigma$ by dropping vertical lines [4]. Figure 1 (right) shows the Hasse diagram of $\mathfrak{W}_3$.

A lower bound $z$ of two elements in a poset $x, y$ satisfies $z \preceq x$ and $z \preceq y$. The greatest lower bound of $x, y$ or infimum is a lower bound of $x, y$ that is greater than or equal to any other lower bound of $x, y$. Infimum, and the analogous concept of supremum are denoted by $x \wedge y$ and $x \vee y$ or $\bigwedge\{x_1, \ldots, x_k\}$ and $\bigvee\{x_1, \ldots, x_k\}$ respectively. Two elements $x, y \in \mathfrak{W}_n$ are consistent if there exists a lower bound in $\mathfrak{W}_n$. Note that consistency is a weaker relation than comparability. For example, $1|2, 3|4$ and $1, 2|3, 4$ are consistent but incomparable while $1|2, 3|4$ and $2|1, 3|4$ are both inconsistent and incomparable. Using the vertical bar notation, two elements are inconsistent iff there exists two items $i, j$ that appear on opposing sides of a vertical bar in $x, y$ i.e. $x = \cdots i|j \cdots$ while $y = \cdots j|i \cdots$. A poset for which $\wedge$ and $\vee$ always exist is called a lattice. Lattices satisfy many useful combinatorial properties - one of which is that they are completely described by the $\wedge$ and $\vee$ operations. While the ranking poset is not a lattice, it may be turned into one by augmenting it with a minimum element $\hat{0}$.

**Proposition 3.** *The union $\tilde{\mathfrak{W}}_n \overset{\text{def}}{=} \mathfrak{W}_n \cup \{\hat{0}\}$ of the ranking poset and a minimum element is a lattice.*

*Proof.* Since $\tilde{\mathfrak{W}}_n$ is finite, it is enough to show existence of $\wedge, \vee$ for pairs of elements [7]. We begin by showing existence of $x \wedge y$. If $x, y$ are inconsistent, there is no lower bound in $\mathfrak{W}_n$ and therefore the unique lower bound $\hat{0}$ is also the infimum $x \wedge y$. If $x, y$ are consistent, their infimum may be obtained as follows. Since $x$ and $y$ are consistent, we do not have a pair of items $i, j$ appearing as $i|j$ in $x$ and $j|i$ in $y$. As a result we can form a lower bound $z$ to $x, y$ by starting with a list of numbers and adding the vertical bars that are in either $x$ or $y$, for example for $x = 3|1, 2, 5|4$ and $y = 3|2|1, 4, 5$ we have $z = 3|2|1, 5|4$. The resulting $z \in \mathfrak{W}_n$, is smaller than $x$ and $y$ since by construction it contains all preferences (encoded by vertical bars) in $x$ and $y$. It remains to show that for every other lower bound $z'$ to $x$ and $y$ we have $z' \preceq z$. If $z'$ is comparable to $z$, $z' \preceq z$ since removing any vertical bar from $z$ results in an element that is not a lower bound. If $z'$ is not comparable to $z$, then both $z, z'$ contain the vertical bars in $x$ and vertical bars in $y$ possibly with some additional ones. By construction $z$ contains only the essential vertical bars to make it a lower bound and hence $z' \prec z$, contradicting the assumption that $z, z'$ are non-comparable. By Proposition 3.3.1 of [7] a poset for which an infimum is always defined and that has a supremum element is necessarily a lattice. Since we just proved that $\wedge$ always exists for $\tilde{\mathfrak{W}}_n$ and $1, \ldots, n = \bigvee \tilde{\mathfrak{W}}_n$, the proof is complete. $\square$

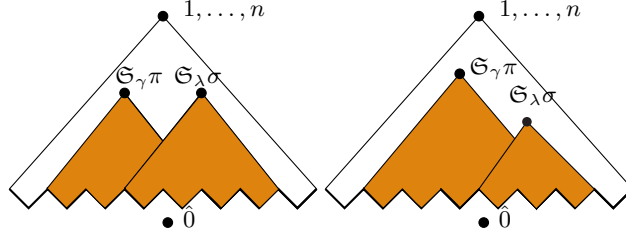

Figure 2: Censored data in the Hasse diagram of $\tilde{\mathfrak{W}}_n$ corresponding to two partial rankings with the same (left) and different (right) number of vertical bars. The two big triangles correspond to the Hasse diagram of Figure 1 (right) with permutations occupying the bottom level.

## 5 Non-Parametric Models on the Ranking Lattice

The censored data approach to partial ranking described by Equation (5) may be generalized to arbitrary probability models $p$ on $\mathfrak{S}_n$. Extending a probability model $p$ on $\mathfrak{S}_n$ to $\tilde{\mathfrak{W}}_n$ by defining it to be zero on $\tilde{\mathfrak{W}}_n \setminus \mathfrak{S}_n$ and considering the partial ranking model

$$g(\mathfrak{S}_\gamma \pi) = \sum_{\sigma \in \mathfrak{S}_\gamma \pi} p(\sigma) = \sum_{\tau \preceq \mathfrak{S}_\gamma \pi} p(\tau), \qquad \tau \in \tilde{\mathfrak{W}}_n. \tag{6}$$

The function $g$, when restricted to partial rankings of the same type $G = \{\mathfrak{S}_\gamma \pi : \pi \in \mathfrak{S}_n\}$ constitutes a distribution over $G$. The relationship between $p$ and $g$ may be more elegantly described through Möbius inversion on lattices: for the functions $p, g : \tilde{\mathfrak{W}}_n \to [0,1]$ defined above we have

$$g(\tau) = \sum_{\tau' \preceq \tau} p(\tau') \quad \text{iff} \quad p(\tau) = \sum_{\tau' \preceq \tau} g(\tau')\mu(\tau', \tau) \qquad \tau, \tau' \in \tilde{\mathfrak{W}}_n \tag{7}$$

where $\mu : \tilde{\mathfrak{W}}_n \times \tilde{\mathfrak{W}}_n \to \mathbb{R}$ is the Möbius function of the lattice $\tilde{\mathfrak{W}}_n$ [7].

For large $n$, modeling partial, rather than full rankings is a computational necessity. It is tempting to construct a statistical model on partial rankings directly without reference to an underlying permutation model, e.g. [1, 6]. However, doing so may lead to contradicting probabilities in the permutation level i.e. there exists no distribution $p$ on $\mathfrak{S}_n$ consistent with the specified values of $g$ at $g(\mathfrak{S}_\gamma \pi)$ and $g(\mathfrak{S}_\lambda \sigma)$, $\gamma \neq \lambda$. Figure 2 illustrates this problem for partial rankings with the same (left) and different (right) number of vertical bars. Verifying that no contradictions exist involves solving a lengthy and complicated set of equations. The alternative we present of starting with a permutation model $p : \mathfrak{S}_n \to \mathbb{R}$ and extending it to $g$ via the Möbius inversion is a simple and effective way of avoiding such lack of coherence.

Identifying partially ranked training data $D = \{\mathfrak{S}_{\gamma_i} \pi_i : i = 1, \dots, m\}$ as censored data, a non-parametric Parzen window estimator based on the Mallows kernel is

$$\hat{p}(\pi) = \frac{1}{m\,\psi(c)} \sum_{i=1}^{m} \frac{1}{|\mathfrak{S}_{\gamma_i}|} \sum_{\tau \in \mathfrak{S}_{\gamma_i} \pi_i} \exp(-c\,d(\pi, \tau)) \qquad \pi \in \mathfrak{S}_n \tag{8}$$

where we used the fact that $|\mathfrak{S}_{\gamma_i} \pi_i| = |\mathfrak{S}_{\gamma_i} e| = |\mathfrak{S}_{\gamma_i}|$, or its censored data extension

$$\hat{g}(\mathfrak{S}_\lambda \pi) = \frac{1}{m\,\psi(c)} \sum_{i=1}^{m} \frac{1}{|\mathfrak{S}_{\gamma_i}|} \sum_{\kappa \in \mathfrak{S}_\lambda \pi} \sum_{\tau \in \mathfrak{S}_{\gamma_i} \pi_i} \exp(-c\,d(\kappa, \tau)) \qquad \mathfrak{S}_\gamma \pi \in \tilde{\mathfrak{W}}_n. \tag{9}$$

Model (8) and its partial ranking extension (9) satisfy requirement 3 in Section 1 since $D$ contains partial rankings of possibly different types. Similarly, by the censored data interpretation of partial rankings, they satisfy requirement 2. Requirement 4 holds as $m, c \to \infty$ by standard properties of the Parzen window estimator. Requirement 5 holds since $\hat{g}$ in (9) restricted to $G = \{\mathfrak{S}_\gamma \pi : \pi \in \mathfrak{S}_n\}$ becomes a consistent model on a much smaller probability space. Requirement 1 is demonstrated in the next section by deriving an efficient computation of (9). In the case of a very large number of items reverting to partial ranking of type $\gamma$ is a crucial element. The coherence between $\hat{p}$, $\hat{g}$ and

the nature of $D$ are important factors in modeling partially ranked data. In the next section we show that even for $n \to \infty$ (as is nearly the case for web-search), the required computation is feasible as it depends only on the complexity of the composition $\gamma$ characterizing the data $D$ and the partial rankings on which $\hat{g}$ is evaluated.

## 6 Efficient Computation and Inversion Combinatorics

Computational efficiency of the inner summations in Equations (8)-(9) is crucial to the practical application of the estimators $\hat{p}, \hat{g}$. By considering how the pairs constituting $i(\tau)$ decompose with respect to certain cosets we can obtain efficient computational schemes for (8),(9).

**Proposition 4.** *The following decomposition of $i(\tau)$ with respect to a composition $\gamma$ holds*

$$i(\tau) = \sum_{k=1}^{r} a_k^{\gamma}(\tau) + \sum_{k=1}^{r} \sum_{l=k+1}^{r} b_{kl}^{\gamma}(\tau) \qquad \forall \tau \in \mathfrak{S}_n \qquad where \tag{10}$$

$$a_k^{\gamma}(\tau) \overset{\text{def}}{=} \left| \left\{ (s,t) \, : \, s < t, \sum_{j=1}^{k-1} \gamma_j < \tau^{-1}(t) < \tau^{-1}(s) \le \sum_{j=1}^{k} \gamma_j \right\} \right| \tag{11}$$

$$b_{kl}^{\gamma}(\tau) \overset{\text{def}}{=} \left| \left\{ (s,t) \, : \, s < t, \sum_{j=1}^{k-1} \gamma_j < \tau^{-1}(t) \le \sum_{j=1}^{k} \gamma_j \le \sum_{j=1}^{l-1} \gamma_j < \tau^{-1}(s) \le \sum_{j=1}^{l} \gamma_j \right\} \right|. \tag{12}$$

*Proof.* First note that by the right invariance of Kendall's tau $d(\tau, \sigma) = i(\tau\sigma^{-1})$, we have $i(\tau) = i(\tau^{-1})$ and we may decompose $i(\tau^{-1})$ instead of $i(\tau)$. The set appearing in the definition of $a_k^{\gamma}(\tau)$ contains all label pairs $(s,t)$ that are inversions of $\tau^{-1}$ and that appear in the $k$-compartment of the decomposition $\gamma$. The set appearing in the definition of $b_{kl}^{\gamma}(\tau)$ contains label pairs $(s,t)$ that are inversions of $\tau^{-1}$ and for which $s$ and $t$ appear in the $l$ and $k$ compartments of $\gamma$ respectively. Since any inversion pair appear in either one or two compartments, the decomposition holds. $\square$

Decomposition (10) is actually a family of decompositions as it holds for all possible compositions $\gamma$. For example, $i(\tau) = 4$ for $\tau = 4|1|3|2 \in \mathfrak{S}_{4-2}\pi = 1,4|2,3$, with inversions $(4,1),(4,3),(4,2),(3,2)$ for $\tau^{-1}$. The first compartment $1,4$ contains the inversion $(4,1)$ and so $a_1^{\gamma}(\tau) = 1$. The second compartment $2,3$ contains the inversion $(3,2)$ and so $a_2^{\gamma}(\tau) = 1$. The cross compartment inversions are $(4,3),(4,2)$ making $b_{12}^{\gamma}(\tau) = 2$. The significance of (10) is that as we sum over all representatives of the coset $\tau \in \mathfrak{S}_{\gamma}\pi$ the cross compartmental inversions $b_{kl}^{\gamma}(\tau)$ remain constant while the within-compartmental inversions $a_k^{\gamma}(\tau)$ vary over all possible combinations. This leads to powerful extensions of Proposition 2 which in turn lead to efficient computation of (8), (9).

**Proposition 5.** *For $\pi \in \mathfrak{S}_n$, $q > 0$, and a composition $\gamma$ we have*

$$\sum_{\tau \in \mathfrak{S}_{\gamma}\pi} q^{i(\tau)} = q^{\sum_{k=1}^{r} \sum_{l=k+1}^{r} b_{kl}^{\gamma}(\pi)} \prod_{s=1}^{r} \prod_{j=1}^{\gamma_s-1} \sum_{k=0}^{j} q^k. \tag{13}$$

*Proof.*

$$\sum_{\tau \in \mathfrak{S}_{\gamma}\pi} q^{i(\tau)} = \sum_{\tau \in \mathfrak{S}_{\gamma}\pi} q^{\sum_{k=1}^{r} a_k^{\gamma}(\tau) + \sum_{k=1}^{r} \sum_{l=k+1}^{r} b_{kl}^{\gamma}(\tau)} = q^{\sum_{k=1}^{r} \sum_{l=k+1}^{r} b_{kl}^{\gamma}(\pi)} \sum_{\tau \in \mathfrak{S}_{\gamma}\pi} q^{\sum_{k=1}^{r} a_k^{\gamma}(\tau)}$$

$$= q^{\sum_{k=1}^{r} \sum_{l=k+1}^{r} b_{kl}^{\gamma}(\pi)} \prod_{s=1}^{r} \sum_{\tau \in \mathfrak{S}_{\gamma_s}} q^{i(\tau)} = q^{\sum_{k=1}^{r} \sum_{l=k+1}^{r} b_{kl}^{\gamma}(\pi)} \prod_{s=1}^{r} \prod_{j=1}^{\gamma_s-1} \sum_{k=0}^{j} q^k.$$

Above, we used two ideas: (i) disconcordant pairs between two different compartments of the coset $\mathfrak{S}_{\gamma}\pi$ are invariant under change of the coset representative, and (ii) the number of disconcordant pairs within a compartment varies over all possible choices enabling the replacement of the summation by a sum over a lower order symmetric group. $\square$

An important feature of (13) is that only the first and relatively simple term $q^{\sum_{k=1}^{r} \sum_{l=k+1}^{r} b_{kl}^{\gamma}(\pi)}$ depends on $\pi$. The remaining terms depend only on the partial ranking type $\gamma$ and thus may be pre-computed and tabulated for efficient computation. The following two corollaries generalize the well known Proposition 2 to arbitrary cosets enabling efficient computation of (8), (9).

| $\lambda \backslash \gamma$ | $(1, n-1)$ | $(1, \cdots, 1, n-t)$ | $(t, n-t)$ |
|---|---|---|---|
| $(1, n-1)$ | $O(1)$ | $O(1)$ | $O(1)$ |
| $(1, \cdots, 1, n-k)$ | $O(k)$ | $O(k+t)$ | $O(k+t)$ |
| $(k, n-k)$ | $O(k)$ | $O(k+t)$ | $O(k+t)$ |

Table 1: Computational complexity for computing Equation (9) for each training example. Notice the independence of the complexity terms from $n$.

**Corollary 1.** $\sum_{\tau \in \mathfrak{S}_\gamma \pi} q^{i(\tau \kappa)} = q^{\sum_{k=1}^r \sum_{l=k+1}^r b_{kl}^\gamma(\pi \kappa)} \prod_{s=1}^r \prod_{j=1}^{\gamma_s - 1} \sum_{k=0}^j q^k \qquad \kappa \in \mathfrak{S}_n$.

*Proof.* Using group theory, it can be shown that the set equality $(\mathfrak{S}_\gamma \pi)\kappa = \mathfrak{S}_\gamma(\pi\kappa)$ holds. As a result, $\sum_{\tau \in \mathfrak{S}_\gamma \pi} q^{i(\tau\kappa)} = \sum_{\tau' \in \mathfrak{S}_\gamma(\pi\kappa)} q^{i(\tau')}$. Proposition 5 completes the proof. $\qquad \square$

**Corollary 2.** *The partial ranking extension corresponding to the Mallows model $p_\kappa$ is*

$$p_\kappa(\mathfrak{S}_\gamma \pi) = \frac{\prod_{s=1}^r \prod_{j=1}^{\gamma_s-1} \sum_{k=0}^j e^{-kc}}{\prod_{j=1}^{n-1} \sum_{k=0}^j e^{-kc}} \, e^{-c\sum_{k=1}^r \sum_{l=k+1}^r b_{kl}^\gamma(\pi\kappa^{-1})} \propto e^{-c\sum_{k=1}^r \sum_{l=k+1}^r b_{kl}^\gamma(\pi\kappa^{-1})}$$

*Proof.* Using Corollary 1 we have

$$p_\kappa(\mathfrak{S}_\gamma \pi) = \sum_{\tau \in \mathfrak{S}_\gamma \pi} p_\kappa(\tau) = \frac{\sum_{\tau \in \mathfrak{S}_\gamma \pi} \exp(-c\, d(\tau, \kappa))}{\sum_{\tau \in \mathfrak{S}_n} \exp(-c\, d(\tau, \kappa))} = \frac{\sum_{\tau \in \mathfrak{S}_\gamma \pi} \exp(-c\, i(\tau\kappa^{-1}))}{\prod_{j=1}^{n-1} \sum_{k=0}^j e^{-kc}}$$

$$= \frac{\sum_{\tau \in \mathfrak{S}_\gamma \pi} (\exp(-c))^{i(\tau\kappa^{-1})}}{\prod_{j=1}^{n-1} \sum_{k=0}^j e^{-kc}} = e^{-c\sum_{k=1}^r \sum_{l=k+1}^r b_{kl}^\gamma(\pi\kappa^{-1})} \frac{\prod_{s=1}^r \prod_{j=1}^{\gamma_s-1} \sum_{k=0}^j e^{-kc}}{\prod_{j=1}^{n-1} \sum_{k=0}^j e^{-kc}}$$

$$\square$$

Despite its daunting appearance, the expression in Corollary 2 can be computed relatively easily. The fraction does not depend on $\pi$ or $\kappa$ and in fact may be considered as a normalization constant that may be easily pre-computed and tabulated. The remaining term is relatively simple and depends on the location parameter $\kappa$ and the coset representative $\pi$. Corollary 2 and Proposition 6 below (whose proof is omitted due to lack of space), provide efficient computation for the estimators (8), (9). The complexity of computing (14) and (8), (9) for some popular partial ranking types appears in Table 1.

**Proposition 6.**

$$\sum_{\sigma \in \mathfrak{S}_\lambda \pi_1} \sum_{\tau \in \mathfrak{S}_\gamma \pi_2} e^{-c\, d(\sigma, \tau)} = \left( \sum_{\tau \in \pi_1 \pi_2^{-1} \mathfrak{S}_\gamma} \prod_{k=1}^r \prod_{l=k+1}^r e^{-c\, b_{kl}^\lambda(\tau)} \right) \left( \prod_{s=1}^r \prod_{j=1}^{\lambda_s-1} \sum_{k=0}^j e^{-kc} \right). \qquad (14)$$

## 7 Applications

Figure 3 (top left) compares the average test log-likelihood between the Mallows model and the non-parametric model with different $c$ as a function of training size averaged over 10 cross validations. We use fully ranked APA election data (rankings are ballots for five APA presidential candidates), and during each iteration, 30% of the examples are randomly selected for testing. The parameters of the Mallows model are estimated by maximum likelihood. The figure illustrates the advantage of using a non-parametric estimator over the parametric Mallows model given enough training data. Also note when $c$ increases, the non-parametric model approaches the empirical histogram thus performing worse for small datasets and better for large datasets. To visualize the advantage of the non-parametric model over the Mallows model we display in Figure 3 (bottom row) their estimated probabilities by scaling the vertices of the permutation polytope proportionally. The displayed polytope has vertices corresponding to rankings of 4 items and whose edges correspond to an adjacent transposition (Kendall's tau distance is the shortest path between two vertices). In this case the four ranked items are movies no. 357, 1356, 440, 25 from the EachMovie dataset containing rankings of 1628 movies. Note how the probabilities assigned by the Mallows model (left) form a unimodal function centered at $2|1|3|4$ while the non-parametric estimator (right) discovers the true modes $2|3|1|4$ and $4|1|2|3$ that were undetected by the Mallows model.

Figure 3 (top right) demonstrates modeling partial rankings of a much larger $n$. We used 10043 rankings from the Jester dataset which contains user rankings of $n = 100$ jokes. We kept the partial ranking type of the testing data fixed at $(5, n - 5)$ and experimented with different censoring of the training data. The figure illustrates the slower consistency rate for fully ranked training data and the statistical benefit in censoring full rankings in the training data. This striking statistical advantage demonstrates the achievement of property 5 in Section 1 and is independent of the computational advantage obtained from censoring the training data.

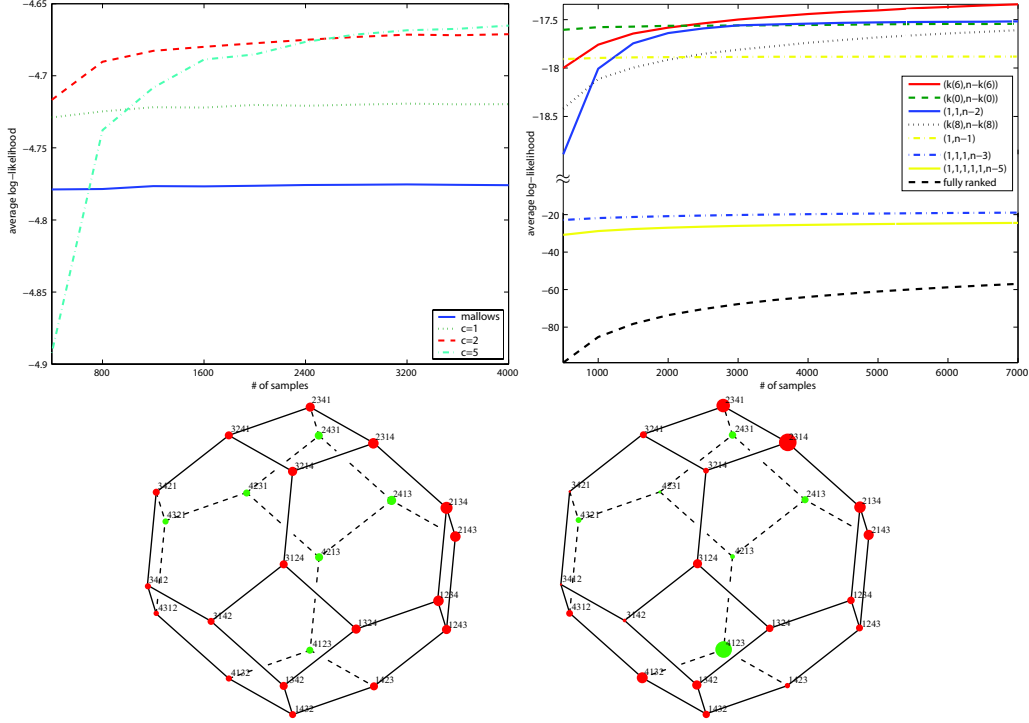

Figure 3: Top row: Average test log-likelihood as a function of the training size: Mallows model vs. non-parametric model for APA election data (left) and non-parametric model with different partial ranking types for Jester data (right). Bottom row: Visualizing estimated probabilities for EachMovie data by permutation polytopes: Mallows model (left) and non-parametric model for $c = 2$ (right).

## 8 Discussion

In this paper, we demonstrate for the first time a non-trivial effective modeling framework satisfying properties 1-5 in Section 1. The key component is our ability to efficiently compute (14) for simple partial ranking types and large $n$. Table 1 indicates the resulting complexity scales up with complexity of the composition $k$ but is independent of $n$ which is critical for modeling practical situations of $k \ll n$ partial rankings. Experiments show the statistical advantage of the non-parametric partial ranking modeling in addition to its computational feasibility.

## References

[1] D. E. Critchlow. *Metric Methods for Analyzing Partially Ranked Data*. Springer, 1986.

[2] M. A. Fligner and J. S. Verducci. Distance based ranking models. *Journal of the Royal Statistical Society B*, 43:359–369, 1986.

[3] M. G. Kendall. A new measure of rank correlation. *Biometrika*, 30, 1938.

[4] G. Lebanon and J. Lafferty. Conditional models on the ranking poset. In *Advances in Neural Information Processing Systems, 15*, 2003.

[5] C. L. Mallows. Non-null ranking models. *Biometrika*, 44:114–130, 1957.

[6] J. I. Marden. *Analyzing and modeling rank data*. CRC Press, 1996.

[7] R. P. Stanley. *Enumerative Combinatorics*, volume 1. Cambridge University Press, 2000.

